# When Does Non-Negative Matrix Factorization Give a Correct Decomposition into Parts?

**David Donoho**
Department of Statistics
Stanford University
Stanford, CA 94305
donoho@stat.stanford.edu

**Victoria Stodden**
Department of Statistics
Stanford University
Stanford, CA 94305
vcs@stat.stanford.edu

## Abstract

We interpret non-negative matrix factorization geometrically, as the problem of finding a simplicial cone which contains a cloud of data points and which is contained in the positive orthant. We show that under certain conditions, basically requiring that some of the data are spread across the faces of the positive orthant, there is a unique such simplicial cone. We give examples of synthetic image articulation databases which obey these conditions; these require separated support and factorial sampling. For such databases there is a generative model in terms of 'parts' and NMF correctly identifies the 'parts'. We show that our theoretical results are predictive of the performance of published NMF code, by running the published algorithms on one of our synthetic image articulation databases.

## 1 Introduction

In a recent article in *Nature* [4], Lee and Seung proposed the notion of non-negative matrix factorization (NMF) as a way to find a set of basis functions for representing non-negative data. They claimed that the notion is particularly applicable to image articulation libraries made up of images showing a composite object in many articulations and poses. They suggested (in the very title of the article) that when used in the analysis of such data, NMF would find the intrinsic 'parts' underlying the object being pictured.

NMF is akin to other matrix decompositions which have been proposed previously, such as positive matrix factorization (PMF) of Juvela, Lehtinen, and Paatero [3], [2] and various minimum-volume transforms used in the analysis of remote-sensing data [1]. Numerous applications of these methods have been attempted [6], [7], [9].

Despite all the literature and discussion of this method, two fundamental questions appear not to have been posed clearly, let alone answered:

- Under what assumptions is the notion of non-negative matrix factorization well-defined, for example is the factorization in some sense *unique*?

- Under what assumptions is the factorization *correct*, recovering the 'right answer'?

In this paper, we develop a geometric view of the setting underlying NMF factorization and derive geometric conditions under which the factorization is essentially unique, so NMF makes sense *no matter what algorithm is being employed*. We then consider those conditions in the setting of image articulation libraries. We describe a class of image libraries which are created by an NMF-style generative model, where different parts have separate support, and where all different combinations of parts are exhaustively sampled. Our theory shows that, in such Separable Factorial Articulation Families, non-negative factorization is effectively unique. In such libraries, NMF will indeed successfully 'find the parts'. We construct such a library, showing a stick figure with four limbs going through a range of various motions, and verify that our theoretical analysis is predictive of the actual performance of the Lee and Seung algorithm on this image library. Our viewpoint also explains relations between NMF and other ideas for obtaining non-negative factorizations and explains why uniqueness and stability may fail under other conditions.

We note that Plumbley [5] has in some sense already validated NMF for datasets which are not only non-negative but which obey an independent components model. However, in our view, this is actually a result about independent components analysis, not NMF. For example, for the kinds of image articulation families where each part is viewed in one of many positions, the underlying exclusion principle – that a certain part can only be present in one particular articulation – guarantees that an ICA model does not apply. And this parts-based setting is exactly the setting for NMF envisioned by Seung and Lee.

## 2   Non-Negative Matrix Factorization

NMF seeks to decompose a non-negative $n \times p$ matrix $X$, where each row contains the $p$ pixel values for one of the $n$ images, into

$$X = A\Psi \tag{1}$$

where $A$ is $n \times r$ and $\Psi$ is $r \times p$, and both $A$ and $\Psi$ have non-negative entries. The rows of $\Psi$, denoted $(\psi_j)_{j=1}^r$, are basis elements in $\mathbf{R}^p$ and the rows of $A$, $(\alpha^i)_{i=1}^n$, belong to $R^r$ and can be thought of as coefficient sequences representing the images in that basis. Recalling that the rows of $X$, $(\mathbf{x}^i)$, are individual images stored as row vectors, the representation takes the form

$$\mathbf{x}^i = \sum_{j=1}^r \alpha_j^i \psi_j.$$

Indexing the pixels by $k = 1, \ldots, p$, non-negativity of $\alpha^i$ and $\psi_j$ can be written as:

$$\psi_j(k) \geq 0, \ \ j = 1, \ldots, r, \ k = 1, \ldots, p; \quad \alpha_j^i \geq 0, \ \ j = 1, \ldots, r, \ i = 1, \ldots, n. \tag{2}$$

It is clear that as a *generative model*, this approach makes sense; each of us can think of some admittedly very simple imaging settings where the scene is composed out of 'standard parts' in a variety of positions, where these are represented by the $\psi_j$ and each image is made by superposing some of those 'parts'. In this setting each part is either present or absent, and the corresponding coefficient is thus positive or zero. An example of this kind will be given in Section 4 below.

What is less clear is whether, when the generative model actually holds and we generate a synthetic dataset based on that model, the NMF matrix factorization of the dataset will yield underlying basis elements which have some connection to the true generative elements. In this paper we investigate this question and exhibit conditions under which NMF will in fact successfully recover the true generative elements.

# 3   Geometric Interpretation of the NMF Setting

We now describe a geometric viewpoint which will help explain the issues involved.

Each image in our database of images can be thought of as a point in a $p$-dimensional space, whose $p$ coordinates are given by the intensity values in each of the $p$ pixels. The fact that image data are non-negative means that every such point lies in the positive orthant $\mathcal{P}$ of $\mathbf{R}^p$.

The factorization $X = A\Psi$ says that there are vectors $\psi_j$ in $\mathbf{R}^p$ such that all the data points $\mathbf{x}^i$ have a representation as non-negative linear combinations of the $\psi_j$. This algebraic characterization has a geometric counterpart.

**Definition.** *The* **simplicial cone** *generated by vectors* $\Phi = (\phi_j)_{j=1}^r$ *is*

$$\Gamma = \Gamma_\Phi = \{\mathbf{x} : \mathbf{x} = \sum_j \alpha_j \phi_j, \alpha_j \geq 0\}.$$

The factorization (1) tells us geometrically that the $(\mathbf{x}^i)$ all lie in the simplicial cone $\Sigma_\Psi$ generated by the $(\psi_j)$.

Now in general, for a given dataset $(\mathbf{x}^i)$, there will be *many* possible simplicial cones containing the points in that dataset. Indeed, if $\Gamma_\Psi$ is a simplicial cone containing the data, and $\Gamma_\Phi$ is another cone containing the first, so that

$$\Gamma_\Psi \subset \Gamma_\Phi,$$

then the corresponding vectors $\Phi = (\phi_j)$ also can furnish a representation of the dataset $(\mathbf{x}^i)$. Now for any simplicial cone, there can always be another cone containing it strictly, so there are an infinite number of factorizations $X = A\Psi$ with non-negative $A$, and various $\Psi$ which are nontrivially different. Hence the constraint $A \geq 0$ is not enough to lead to a well-defined notion.

However, the geometric viewpoint we are developing does not so far include the positivity constraint $\Psi \geq 0$ on the generating vectors of the simplicial cone. Geometrically, this constraint demands that the simplicial cone $\Gamma_\Psi$ lies inside the positive orthant $\mathcal{P}$. Can we obtain uniqueness with this extra constraint?

Not if the data values are strictly positive, so that

$$X_{i,k} \geq \epsilon > 0 \qquad \forall i, k. \tag{3}$$

Geometrically, this condition places the data points $\mathbf{x}^i$ well inside the interior of the positive orthant $\mathcal{P}$. It is then evident by visual inspection that there will be *many* simplicial cones containing the data. For example, $\mathcal{P}$ itself is a simplicial cone, and it contains the data points. However, many other cones will also contain the data points. Indeed, for $\delta > 0$ consider the collection of vectors $\Phi^\delta$ with individual vectors

$$\phi_j^\delta = e_j + \delta \mathbf{1}$$

where $e_j$ denotes the usual vector in the standard basis, and $\mathbf{1}$ denotes the vector of all ones. Then, for $\delta < \epsilon$, the cone $\Gamma_{\Phi^\delta}$ also contains all the data points. Geometrically $\Gamma_{\Phi^\delta}$ is a dilation of the positive orthant that shrinks it slightly towards the main diagonal. Since the positivity constraint (3) places all the data well inside the interior of the positive orthant, for slight enough shrinkage it will still contain the data.

It follows from the geometric-algebraic correspondence that under the strict positivity condition (3), there are many distinct representations $X = A\Psi$ where $A \geq 0$ and $\Psi \geq 0$.

In short, we must look for situations where the data do not obey strict positivity in order to have uniqueness.

## 4   An Example of Uniqueness

When we take the non-negativity constraint on the generating elements (the extreme rays of the simplicial cone) into account, it can happen that there will only be one simplicial cone containing the data. This is completely clear if the data somehow 'fill out' the positive orthant. What is perhaps surprising is that uniqueness can hold even when the data only 'fill out' a proper subset of the positive orthant.

Here is an example of how that can occur. Consider the 'ice-cream cone'

$$C = \{\mathbf{x} : \mathbf{x}'\mathbf{1} \geq \sqrt{p-1}||\mathbf{x}||\}$$

where $p$ is again the dimensionality of the dataspace.

**Lemma 1.** *There is a unique simplicial cone which both contains $C$ and is itself contained in the positive orthant.*

Indeed that unique cone is $\mathcal{P}$ itself; no simplicial cone contained inside $\mathcal{P}$ contains all of $C$!

To give a full proof, we introduce notions from the subject of convex duality [8]. Associated with the primal domain of points $x$ we have been dealing with so far, there is also the dual domain of linear functionals $\xi$ acting on points $x$ via $\xi'x$. If we have a convex set $C$, its dual $C^*$ is defined as a collection of linear functionals which are positive on $C$:

$$C^* = \{\xi : \xi'\mathbf{x} \geq 0 \quad \forall \mathbf{x} \in C\}$$

The following facts are easily verified:

**Lemma 2.**

- *If $K$ is closed and convex then $(K^*)^* = K$.*
- *The dual of a simplicial cone with $p$ linearly independent generators, is another simplicial cone with $p$ generators.*
- *The positive orthant is self-dual: $\mathcal{P}^* = \mathcal{P}$.*
- *Duality reverses set inclusion:*

$$B \subset C \implies C^* \subset B^*. \tag{4}$$

We also need

**Definition.** *Given a pointset $(\mathbf{x}^i)$, its* **conical hull** *is the simplicial hull generated by the vectors $(\mathbf{x}^i)$ themselves.*

Let $\mathcal{X}$ be the conical hull of a pointset. An abstraction of the NMF problem is:

**Primal-Simplicial-Cone**$(r, \mathcal{X})$ *Find a simplicial cone with $r$ generators contained in $\mathcal{P}$ and containing $\mathcal{X}$.*

Consider now a problem in the dual domain, posed with reversed inclusions:

**Dual-Simplicial-Cone**$(r, \Xi)$ *Find a simplicial cone with $r$ generators contained in $\Xi$ and containing $\mathcal{P}$ .*

The two problems are indeed dual:

**Lemma 3.** *Every solution to Primal-Simplicial-Cone$(r, \mathcal{X})$ is dual to a solution of Dual-Simplicial-Cone$(r, \mathcal{X}^*)$, and vice-versa.*

**Proof.** This is effectively the invocation of 'reversal of inclusion under duality' (4). Suppose we find a simplicial cone $\Gamma$ obeying

$$\mathcal{X} \subset \Gamma \subset \mathcal{P}.$$

Then (4) says that

$$\mathcal{P}^* \subset \Gamma^* \subset \mathcal{X}^*,$$

and so a solution to the primal solves the dual. In the other direction, if we find a simplicial cone $\Gamma^*$ obeying

$$\mathcal{P}^* \subset \Gamma^* \subset \mathcal{X}^*$$

then we have by (4)

$$(\mathcal{X}^*)^* \subset (\Gamma^*)^* \subset (\mathcal{P}^*)^*;$$

we simply apply $(K^*)^* = K$ three times to see that a solution to the dual corresponds to a solution to the primal. QED

Our motivation in introducing duality is to see something we couldn't in the primal: we can see that *even if $\mathcal{X}$ is properly contained in $\mathcal{P}$, there can be a unique simplicial hull for $\mathcal{X}$ which lies inside $\mathcal{P}$.*

This follows from a simple observation about simplicial cones contained in convex cones.

**Definition.** *An **extreme ray** of a convex cone $\Gamma$ is a ray $R_{\mathbf{x}} = \{a\mathbf{x} : a \geq 0\}$ where $\mathbf{x} \in \Gamma$ cannot be represented as a proper convex combination of two points $\mathbf{x}_0$ and $\mathbf{x}_1$ which belong to $\Gamma$ but not $R_{\mathbf{x}}$.*

For example, a simplicial cone with $r$ linearly independent generators has $r$ extreme rays; each ray consists of all positive multiples of one generator.

**Lemma 4.** *Suppose that $\Gamma$ and $G$ are convex cones, that $\Gamma \subset G \subset R^r$, that $\Gamma$ is a simplicial cone with $r$ generators and that $G$ intersects $\Gamma$ in exactly $r$ rays which are extreme rays of $G$. Then (a) these rays are also extreme rays of $\Gamma$ and (b) no simplicial cone with $r$ generators $\Gamma' \neq \Gamma$ can satisfy $\Gamma \subset \Gamma' \subset G$.*

**Proof.** (a) Since the rays in question are extreme rays of $G$, which contains $\Gamma$, they are also extreme rays of $\Gamma$. (b) Any simplicial cone $\Gamma'$ with $r$ generators and lying 'in between' $\Gamma$ and $G$ would have to also intersect $G$ in the same $r$ rays as $\Gamma$ does. Those $r$ rays would also have to be extreme rays for $\Gamma'$, because they are extreme rays for $G$, which by hypothesis contains $\Gamma'$. But a simplicial cone with $r$ generators is completely determined by its $r$ extreme rays. As $\Gamma$ and $\Gamma'$ have the same extreme rays, $\Gamma = \Gamma'$. QED

We can now prove Lemma 1. Recall the cone $C$ defined above. Its dual is

$$C^* = \{\xi : \xi' \mathbf{1} \geq ||\xi||\}$$

Note (a) that every boundary ray of $C^*$ is extreme; and (b) that $C^*$ intersects $\mathcal{P}^*$ on the $n$ unit vectors $e_j$. So by Lemma 4, $\mathcal{P}^*$ uniquely solves the Dual-Simplicial-Cone$(n, C^*)$ problem and $\mathcal{P}$ solves the Primal-Simplicial-Cone$(n, C)$ problem uniquely. QED.

## 5 Uniqueness for Separable Factorial Articulation Families

We now describe families of articulated images which have at least a few 'realistic' features, and which, because of the relevant convex geometry, offer an essentially unique NMF.

The families of images we have in mind consist of black-and-white images with $P$ parts, each exercised systematically through $A$ articulations. As an illustration, Figure 1 shows some sample images from the Swimmer dataset, which depicts a figure with four moving parts (limbs), each able to exhibit four articulations (different positions).

**Definition.** A **Separable Factorial Articulation Family** is a collection $X$ of points $\mathbf{x}$ obeying these rules:

**[R1]** *Generative Model.* Each image $\mathbf{x}$ in the database has a representation

$$\mathbf{x} = \sum_{q=1}^{P} \sum_{a=1}^{A} \alpha_{q,a} \psi_{q,a}$$

where the generators $\psi_{q,a} \in R^p$ obey the non-negativity constraint $\psi_{q,a} \geq 0$ along with the coefficients $\alpha_{q,a} \geq 0$. We speak of $\psi_{q,a}$ as the $q$'th part in the '$a$'-th articulation.

**[R2]** *Separability.* For each $q, a$ there exists a pixel $k_{q,a}$ such that

$$\psi_{q',a'}(k_{q,a}) = 1_{\{a=a',q=q'\}} \tag{5}$$

I.e. each part/articulation pair's presence or absence in the image is indicated by a certain pixel associated to that pair.

**[R3]** *Complete Factorial Sampling.* The dataset contains all $A^P$ images in which the $P$ parts appear in all combinations of $A$ articulations.

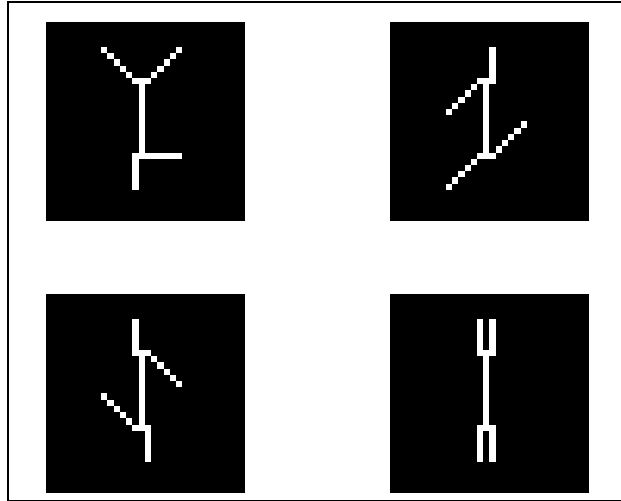

Figure 1: *Sample images from the* Swimmer *database depicting four stick figures with four limbs; the panels illustrate different articulations of the limbs.*

The Swimmer dataset obeys these rules except for one disagreement: every image contains an invariant region (the torso). As it turns out this is of small importance.

We note that assumption **[R2]** forces the generators $\psi_{q,a}$ to be linearly independent, which forces $p > A \cdot P$. Consequently, the linear span of the generators is some subspace $V \subset R^p$.

**Theorem 1.** *Given a database obeying rules* **[R1]**-**[R3]**, *there is a unique simplicial hull with $r = A \cdot P$ generators which contains all the points of the database, and is contained in $\mathcal{P} \cap V$.*

Since the generative model **[R1]** implies that a particular simplicial hull with a specific choice of $r$ generators contains the dataset, and a successful application of NMF also gives a simplicial hull with $r$ generators containing the dataset, and the theorem says these must be the same hull, in this setting NMF recovers the generative model. Formally,

**Corollary.** *Let $X$ be generated by rules* **[R1]**-**[R3]**. *Any factorization obeying (1) and (2) must recover the correct generators $(\psi_{q,a})$ modulo permutation of labels and rescaling.*

# 6  Proof of Theorem 1.

We need to introduce the notion of duality relative to a vector space $V \subset R^p$. In the case of $V \equiv R^p$ this is just the notion of duality already introduced. Suppose that we have a set $K \subset V$; its relative dual $K^v$ is the set of linear functionals $\xi$ which, viewed as members of $R^p$ also belong to $V$, and which obey $\xi'\mathbf{x} \geq 0$ for $\mathbf{x} \in K$. In effect, the relative dual is the ordinary dual taken within $V$ rather than $R^p$. As a result, all the properties of Lemma 2 hold for relative duality provided we talk about sets which are subsets of $V$; e.g. $(K^v)^v = K$ if $K$ is a closed convex subset of $V$.

Define $\mathcal{P}_V = V \cap \mathcal{P}$; this is a simplicial cone in $V$ with $r$ generators.

Let again $\mathcal{X}$ denote the conical hull of $X = (\mathbf{x}^i)$ and suppose that every $(r-1)$-dimensional face of $\mathcal{P}_V$ contains $r-1$ linearly independent points from $X$. Since the face of a cone is a linear subspace, the face is uniquely determined by these $r-1$ points. The face is part of a supporting hyperplane to $\mathcal{P}_V$ which is also a supporting hyperplane to $\mathcal{X}$. The supporting hyperplane defines a point $\xi \in V$ which is in common between the duals $\mathcal{P}_V^v$ and $\mathcal{X}^v$. Similar statements hold for all the $r$ different $(r-1)$-faces of $\mathcal{P}_V$. But more is true. Because of the linear independence mentioned above, the different supporting hyperplanes in primal space correspond in fact to extreme rays in dual space – extreme rays for both $\mathcal{P}_V^v$ and $\mathcal{X}^v$. As this is true for all $r$ of the $(r-1)$-dimensional faces, we are in a position to apply Lemma 4 with $G = \mathcal{X}^v$ and $\Gamma = \mathcal{P}_V^v$. This gives the conclusion that $\mathcal{P}_V^v$ is the unique simplicial cone with $r$ generators contained in $\mathcal{X}^v$ and containing $\mathcal{P}_V^v$. Theorem 1 then follows by duality.

It remains to establish the assumption about existence of $r-1$ linear independent points on each $(r-1)$-face. The faces of $\mathcal{P}_V$ are exactly the $r$ different subspaces

$$F_{q,a} = \{\mathbf{x} \in V : \alpha_{q,a} = 0\}.$$

By the Complete Factorial Sampling assumption **[R3]**, there are $A^{P-1}(A-1)$ points of $X$ in such a face. Define, for each $(q', a') \neq (q, a)$,

$$\phi_{q',a';q,a} = Ave\{\mathbf{x} \in X : \alpha_{q,a} = 0, \alpha_{q',a'} = 1\}.$$

There are $r-1$ such terms, one for each part/articulation pair besides $(q, a)$. By the Separability assumption **[R2]**:

$$\phi_{q',a';q,a}(k_{q'',a''}) = 1_{\{q'=q'',a'=a''\}}.$$

Hence the $(\phi_{q',a';q,a} : (q', a') \neq (q, a))$ are linearly independent. At the same time,

$$\phi_{q',a';q,a}(k_{q,a}) = 0$$

so that each $\phi_{q',a';q,a} \in F_{q,a}$. Hence we have the required linearly independent subset in each face. QED

# 7  Empirical Verification

We built the `Swimmer` image library of 256 32×32 images. Each image contains a 'torso' of 12 pixels in the center and four 'arms' of 6 pixels that can be in one of 4 positions. All combinations of all possible arm positions gives us 256 images. See Figure 1 for examples.

This collection of images has four 'parts'. It deviates slightly from the rules [R1]-[R5] because there is an invariant region (the torso). Figure 2 shows that the 16 different part/articulation pairs are properly resolved, but that the torso is not properly resolved.

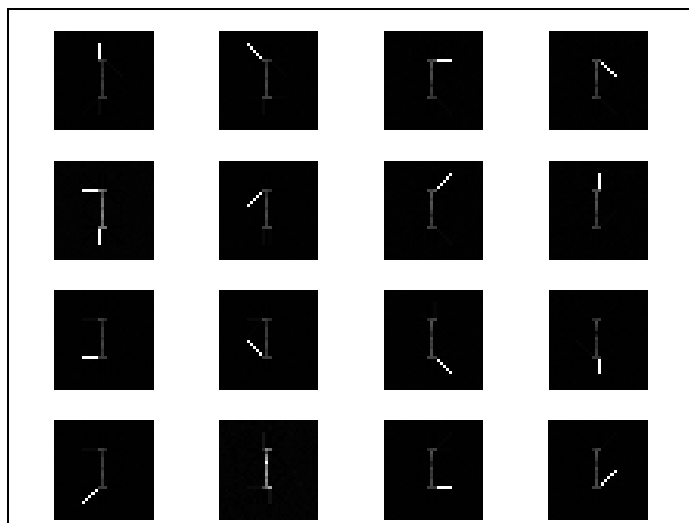

Figure 2: *NMF Generators recovered from* Swimmer *database. The 16 images shown agree well with the known list of generators (4 'limbs' in 4 positions each). The presence of the torso (i.e. an invariant region) violates our conditions for a Factorial Separable Articulation Library, and, not unexpectedly, ghosts of the torso contaminate several of the reconstructed generators. Lee and Seung's code [4] was used.*

## Acknowledgments

This work was partially supported by NSF grants DMS-0077261, DMS-0140698, and ANI-008584 and a contract from DARPA ACMP. We would like to thank Aapo Hyvärinen for numerous helpful discussions.

## References

[1] M. Craig. Minimum-volume transforms for remotely sensed data. *IEEE Transactions on Geoscience and Remote Sensing*, 32(3):542-552, May 1994.

[2] M. Juvela, K. Lehtinen, and P. Paatero. The use of positive matrix factorization in the analysis of molecular line spectra from the thumbprint nebula. In D. P. Clemens and R. Barvainis, editors, *Clouds, Cores, and Low Mass Stars*, volume 65 of *ASP Conference Series*, 176-180, 1994.

[3] M. Juvela, K. Lehtinen, and P. Paatero. The use of positive matrix factorization in the analysis of molecular line spectra. *MNRAS*, 280:616-626, 1996.

[4] D. Lee and S. Seung. Learning the parts of objects by non-negative matrix factorization. *Nature*, 401:788-791, 1999.

[5] M. Plumbley. Conditions for nonnegative independent components analysis. *Signal Processing Letters, IEEE*, 9(6):177-180, 2002.

[6] A. Polissar, P. Hopke, W. Malm, and J. Sisler. Atmospheric aerosol over alaska 1. spatial and seasonal variability. *Journal of Geophysical Research*, 103(D15):19035-19044, August 1998.

[7] A. Polissar, P. Hopke, W. Malm, and J. Sisler. Atmospheric aerosol over alaska 2. elemental composition and sources. *Journal of Geophysical Research*, 103(D15):19045-19057, August 1998.

[8] R. T. Rockefellar. *Convex Analysis*, Princeton University Press, 1970.

[9] W. Size. *Use and Abuse of Statistical Methods in the Earth Sciences*, chapter 3, pages 33-46. Oxford University Press, 1987.
